# Further Studies of a Model for the Development and Regeneration of Eye-Brain Maps

**J.D. Cowan & A.E. Friedman**
Department of Mathematics, Committee on
Neurobiology, and Brain Research Institute,
The University of Chicago, 5734 S. Univ. Ave.,
Chicago, Illinois 60637

## Abstract

We describe a computational model of the development and regeneration of specific eye-brain circuits. The model comprises a self-organizing map-forming network which uses local Hebb rules, constrained by (genetically determined) molecular markers. Various simulations of the development and regeneration of eye-brain maps in fish and frogs are described, in particular successful simulations of experiments by Schmidt-Cicerone-Easter; Meyer; and Yoon.

## 1 INTRODUCTION

In a previous paper published in last years proceedings (Cowan & Friedman 1990) we outlined a new computational model for the development and regeneration of eye-brain maps. We indicated that such a model can simulate the results of a number of the more complicated surgical manipulations carried out on the visual pathways of goldfish and frogs. In this paper we describe in more detail some of these experiments, and our simulations of them.

### 1.1 EYE-BRAIN MAPS

We refer to figure 1 from the previous paper which shows the retinal map found in the optic lobe or tectum of a fish or frog. The map is topological, i.e.; neighborhood

relationships in the retina are preserved in the optic tectum. As is well-known nearly 50 years ago Sperry (1944) showed that such maps are quite precise and specific, in that maps (following optic nerve sectioning and eye rotation) regenerate in such a way that optic nerve fibers reconnect, more or less, to their previous tectal sites. Some 20 years ago Gaze and Sharma (1970) and Yoon (1972) found evidence for plasticity in the expanded and compressed "maps" which regenerate following eye and brain lesions in goldfish. There are now many experiments which indicate that the regeneration of connections involves both specificity and plasticity.

## 1. 2. EXPANDED MAPS

Such properties are seen in a series of more complicated experiments involving the expansion of a half-eye map to a whole tectum. These experiments were carried out by Schmidt, Cicerone and Easter (1978) on goldfish, in which following the expansion of retinal fibers from a half-eye over an entire (contralateral) tectum, and subsequent sectioning of the fibers, diverted retinal fibers from the other (intact) eye are found to expand over the tectum, as if they were also from a half-eye. This has been interpreted to imply that the tectum has no intrinsic positional markers to provide cues for incoming fibers, and that all its subsequent markers come from the retina (Chung & Cooke, 1978). However Schmidt et.al. also found that the diverted fibers also map normally. Figure 4 of the previous paper shows the result.

## 1. 3. COMPRESSED MAPS

Compression is found in maps from entire eyes to ablated half tecta (Gaze & Sharma, 1970; Sharma & Gaze, 1971; Yoon, 1972). There has been considerable controversy concerning the results. Recently Meyer (1982) has shown that although electrophysiological techniques seem to provide evidence for smoothly expanded and compressed maps, autoradiographic techniques do not. Instead of a smooth map there are *patches*, and in many cases no real expansion or compression is seen in irradiated sections, at least not initially. An experiment by Yoon (1976) is relevant here. Yoon noticed that in the early stages of map formation under such conditions, the map is normal. Only after some considerable time does a compressed map form. However if the fibers are sectioned (cut) and allowed to regenerate a second time, compression is immediate. This result has been challenged (Cook, 1979), but it was subsequently confirmed by Schmidt (1983).

## 1. 4. MISMATCHED MAPS

In mismatch experiments, a half retina is confronted with an inappropriate half tectum. In Yoon's classic "mismatch" experiment (Yoon, 1972) fibers from a half-eye fragment are confronted with the "wrong" half-tectum: the resulting map is normally oriented, even though this involves displacement of retinal fibers from near the tectal positions they normally would occupy.

About 12 years ago Meyer (1979) carried out another important mismatch experiment in which the left half of an eye and its attached retinal fibers were surgically removed, leaving an intact normal half-eye map. At the same time the right half the other eye and its attached fibers were removed, and the fibers from the remaining half eye were allowed to innervate the tectum with the left-half eye map. The result is shown in figure 5 of our previous paper. Fibers from the right half-retina, labelled 1 through 5, would normally make contact with the corresponding tectal neurons. Instead they make contact with neurons 6 through 10, but in a *reversed* orientation. Meyer interprets this result to mean that optic nerve fibers show a tendency to aggregate with their nearest *retinal* neighbors.

## 2 THE MODEL

We introduced our model in last year's NIPS proceedings (Cowan & Friedman 1990). We here repeat some of the details. Let $s_{ij}$ be the strength or weight of the synapse made by the ith retinal fiber with the jth tectal cell. Then the following system of differential equations expresses the changes in $s_{ij}$:

$$\dot{s}_{ij} = \lambda_j + c_{ij} [\mu_{ij} + (r_i - \alpha)t_j] s_{ij}$$
$$- \tfrac{1}{2} s_{ij} (T^{-1}\textstyle\sum_i + R^{-1}\textstyle\sum_j)\{\lambda_j + c_{ij} [\mu_{ij} + (r_i - \alpha)t_j] s_{ij}\} \qquad (1)$$

where $i = 1, 2, ...., N_r$, the number of retinal ganglion cells and $j = 1, 2, ...., N_t$, the number of tectal neurons, $c_{ij}$ is the "stickiness" of the ijth contact, $r_i$ denotes retinal activity and $t_j = \sum_i s_{ij} r_i$ is the corresponding tectal activity, and $\alpha$ is a constant measuring the rate of receptor destabilization (see Whitelaw & Cowan (1981) for details). In addition both retinal and tectal elements have fixed lateral inhibitory contacts. The dynamics described by eqn.1 is such that both $\sum_i s_{ij}$ and $\sum_j s_{ij}$ tend to constant values T and R respectively, where T is the total amount of tectal receptor material available per neuron, and R is the total amount of axonal material available per retinal ganglion cell: thus if sij increases anywhere in the net, other synapses made by the ith fiber will decrease, as will other synapses on the jth tectal neuron. In the current terminology, this process is referred to as "winner-take-all".

In addiiton $\lambda_j$ represents a general nonspecific growth of retinotectal contacts, presumed to be controlled and modulated by nerve growth factor (Campenot, 1982). Recent observations (Davies *et.al.*, 1987) indicate that the first fibers to reach a given target neuron stimulate it to produce NGF, which in turn causes more fiber growth. We therefore set $\lambda_j = T^{-1}\sum_i s_{ij}\lambda$ where $\lambda$ is a constant. $\sum_i s_{ij}$ is the instantaneous value of receptor material used to make contacts, and T is the total amount available, so $\lambda_j \to \lambda$ as the jth neuron becomes innervated. The coefficient $\mu_{ij}$ represents a postulated random depolarization which occurs at synapses due to the quantal release of neurotransmitter-- the analog of end-plate potentials (Walmsley *et.al.*, 1987). Thus even if $r_i = 0$, map formation can still occur. However the resulting maps are not as sharp as those formed in

the presence of retinal activity.  Of course if $\mu_{ij} = 0$, as might be the case if $\alpha$-bungarotoxin is administered, then    $\dot{s}_{ij} = \lambda_j(1- s_{ij})$ and $s_{ij} \to 1$, i.e.; all synapses of equal strength.

It is  the coefficients $c_{ij}$. which determine the nature of the solution to eqn.1. These coefficients express the contact adhesion strengths of synapses.  We suppose that such adhesions are generated by fixed distributions of molecules embedded in neural surface membranes. We postulate that the tips of retinal axons and the surfaces of tectal cells display at least two molecular species, labelled a and b, such that $c_{ij} = \sum \xi_{ab}a_ib_j$ and the sum is over all possible combinations aa, ab etc. A number of possibilities exist in the choice of $\xi_{ab}$ and of the spatial distribution of a and b.  One possibility that is consistent with most of the  assays which have been carried out (Trisler & Collins (1987), Bonhoffer and Huff (1980), Halfter, Claviez & Schwarz (1981), Boenhoffer & Huff (1985)) is $\xi_{aa} = \xi_{bb} > 0 > \xi_{ab} = \xi_{ba}$ in which each species prefers itself and repels the other, the so-called homophilic  case, with $a_i$ and $b_i$ as shown in figure 1.

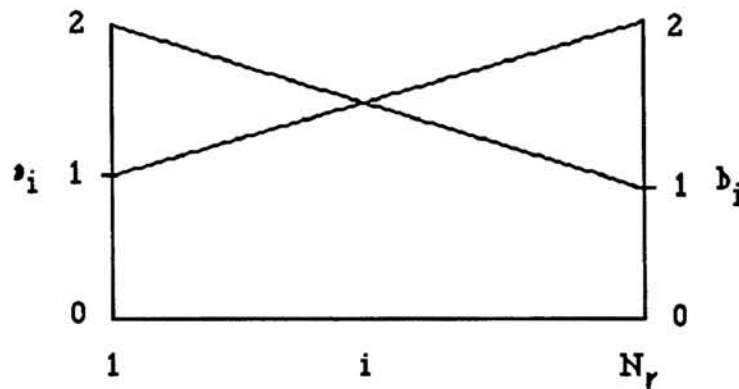

**Figure 1:** Postulated distribution of sticky molecules in the retina.  A similar distribution is supposed to exist in the tectum.

The mismatch and compound eye experiments indicate that  map formation depends in part on a tendency for fibers to stick to their retinal neighbors, in addition to their tendency to stick to tectal cell surfaces.  We therefore append to $c_{ij}$ the term $\sum'_k \bar{s}_{kj} f_{ik}$ where $\bar{s}_{kj}$ is a local average of $s_{kj}$ and its nearest tectal neighbors, where $f_{ik}$ measures themutual stickiness of the ith and kth retinal fibers, and where $\sum'_k$ means $\sum_{k \neq i}$. Fig. 2 shows the postulated form of $f_{ik}$.  {Again we suppose this stickiness is produced by the interaction of two molecular species etc.; specifically theneural contact adhesion molecules  (nCAM) of the sort discovered by Edelman (1983)which seem to mediate the fiber-fiber adhesion observed in tissue cultures by Boenhoffer & Huff (1985), but we do not go into the details}.

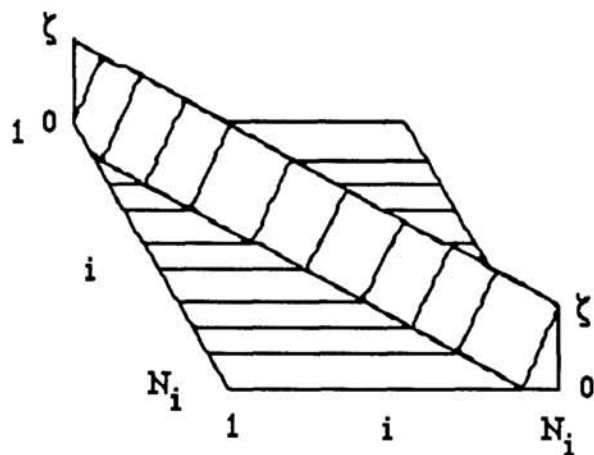

**Figure 2:** The $f_{ik}$ surface. Retinal fibers are attracted only to themselves or to their immediate retinal neighbors.

Meyer's mismatch experiment also indicate that existing fiber projections tend to exclude other fibers, especially inappropriate ones, from innervating occupied areas. One way to incorporate such geometric effects is to suppose that each fiber which establishes contact with a tectal neuron *occludes* tectal markers there by a factor proportional to its synaptic weight $s_{ij}$. Thus we subtract from the coefficient $c_{ij}$ a fraction proportional to $T^{-1} \sum_k' s_{kj}$.

With the introduction of occlusion effects and fiber-fiber interactions, it becomes apparent that *debris* in the form of degenerating fiber fragments adhering to tectal cells, following optic nerve sectioning, can also influence map formation. Incoming nerve fibers can stick to debris, and debris can occlude markers. There are in fact four possibilities: debris can occlude tectal markers, markers on other debris, or on incoming fibers; and incoming fibers can occlude markers on debris. All these possibilities can be included in the dependence of $c_{ij}$ on $s_{ij}$, $s_{kj}$ etc. Note that such debris is supposed to decay, and eventually disappear.

## 3 SIMULATIONS

The model which results from all these modifications and extensions is much more complex in its mathematical structure than any of the previous models. However computer simulation studies show it to be capable of correctly reproducing the observed details of almost all the experiments cited above. For purposes of illustration we consider the problem of connecting a line of $N_r$ retinal cells to a line of $N_t$ tectal cells. The resulting maps can then be represented by two-dimensional matrices, in which the area of the square at the ijth intersection represents the weight of the synapse between the ith retinal fiber and the jth tectal cell. The normal retino-tectal map is represented by large squares along the matrix diagonal., (see Whitelaw & Cowan (1981) for terminology and further details).

## 3.1 THE SCHMIDT ET. AL. EXPERIMENT

Figure 3, for example shows a simulation of the retinal "induction" experiments of Schmidt *et.al*. This simulation generated both an expanded map and a nearly normal patch, interacting to form patches. These effects occur because some incoming retinal fibers stick to debris left over from the previous expanded map, and other fibers stick to non-occluded tectal markers. The fiber-fiber markers control the regeneration of the expanded map, whereas the retino-tectal markers control the formation of the nearly normal map.

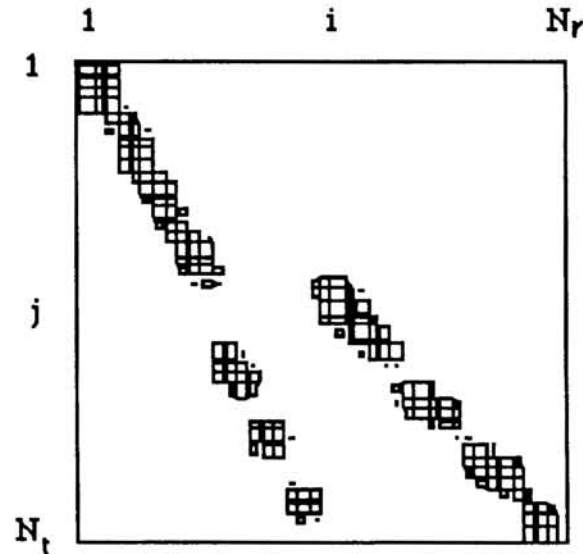

**Figure 3:** Simulation of the Schmidt et.al. retinal induction experiment. A nearly normal map is intercalated into an expanded map.

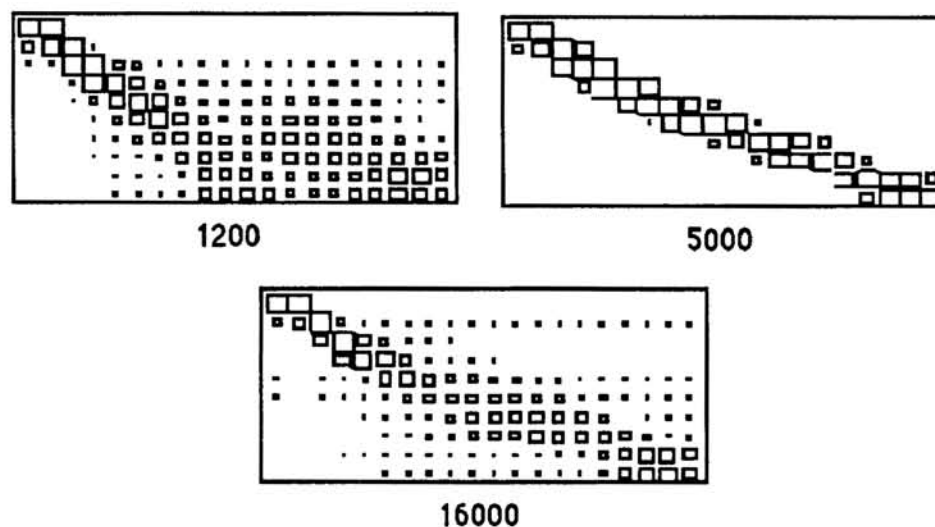

**Figure 4:** Simulation of the Yoon second compression experiment (see text for details).

## 3.2 THE YOON SECOND COMPRESSION EXPERIMENT

Yoon's demonstration of immediate second compression can also be simulated. Figure 4 shows details of the simulation. At an early stage just after the first cut, both a normal and a compressed map are forming. The normal map eventually disappears, leaving only a compressed map. After the second cut however, a compressed map forms immediately. Again it is the debris which carries fiber-fiber markers that control map formation.

## 3.3 THE MEYER MISMATCH EXPERIMENT

It is evident that fiber-fiber interactions are important in controlling map formation. The Meyer mismatch experiment shows this quite clearly. A simulation of this experiment also shows the effect. If $f_{ik}$, the mutual stickiness of neighboring fibers is not strong enough, retino-tectal markers dominate, and the mismatched map forms with normal polarity. However if $f_{ik}$ is large enough, Meyer's result is found, the mismatched map forms with a reversed polarity. Figure 5 shows the details.

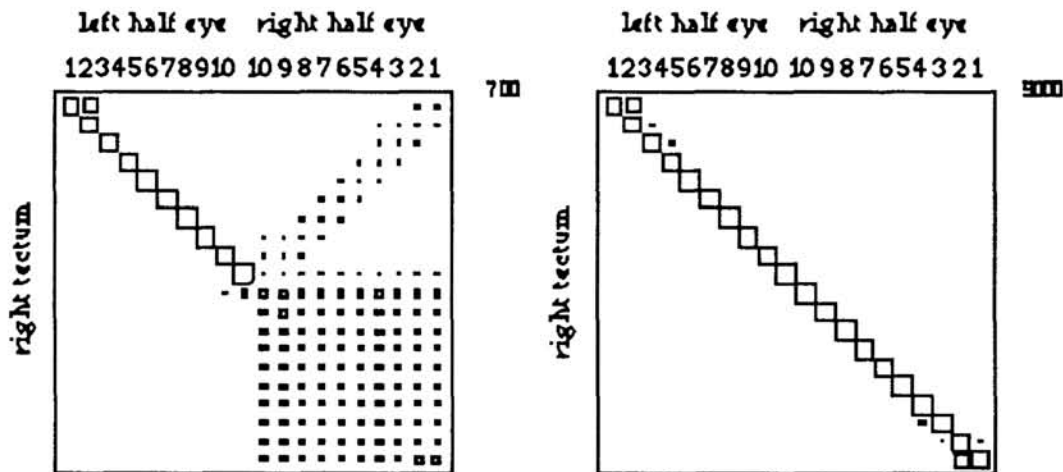

**Figure 5:** Simulation of the Meyer mismatch experiment (see text for details).

# 4 CONCLUSIONS

The model we have outlined generates correctly oriented retinotopic maps. It permits the simulation of a large number of experiments, and provides a consistent explanation of almost all of them. In particular it shows how the apparent induction of central markers by peripheral effects, as seen in the Schmidt et. al., can be produced by the effects of debris, as can Yoon's observations of immediate second compression. Affinity markers are seen to play a key role in such effects, as they do in the polarity reversal seen in Meyer's experiment.

In summary much of the complexity of the many regeneration experiments which have been carried out in the last fifty years can be understood in terms of the effects produced by contact adhesion molecules with differing affinities, acting to control an activity-dependent self-organizing mechanism.

## Acknowledgements

We thank The University of Chicago Brain Research Foundation for partial support of this work.

## References

Boenhoffer, F. & Huf, J. (1980), Nature, **288**, 162-164.; (1985), Nature, **315**, 409-411.

Campenot, R.B. (1982), Develop. Biol., **93**, 1.

Chung, S.-H. & Cooke, J.E. (1978), Proc. Roy. Soc. Lond. B *201*, 335-373.

Cowan, J.D. & A.E. Friedman (1990) Advances in NIPS, **2**, Ed. D.S. Touretzky, Morgan-Kaufmann, 92-99.

Cook, J.E. (1979), J. Embryol. exp. Morphol., **52**, 89-103.

Davies, A.M., Bandtlow, C., Heumann, R, Korsching, S., Rohrer, H. & Thoenen, H. (1987), Nature, **326**, 353-358.

Edelman, G.M., (1983), Science, **219**, 450-454.

Gaze, R.M. & Sharma, S.C. (1970), Exp. Brain Res., 10, 171-181.

Halfter, W., Claviez, M. & Schwarz, U. (1981), Nature, **292**, 67- 70.

Meyer, R.L. (1979), Science, **205**, 819-821; (1982), Curr. Top. Develop. Biol., **17**, 101-145.

Schmidt, J.T. (1983), J. Embryol. exp. Morphol., **77**, 39-51.

Schmidt, J.T., Cicerone, C.M. & Easter, S.S. (1978), J. Comp. Neurol., **177**, 257-288.

Sharma, S.C. & Gaze, R.M. (1971), Arch. Ital. Biol., **109**, 357-366.

Sperry, R.W. (1944), J. Neurophysiol., **7**, 57-69.

Trisler, D. & Collins, F. (1987), Science, **237**, 1208-1210.

Walmsley, B., Edwards, F.R. & Tracey, D.J. (1987), J. Neurosci., **7**, *4*, 1037-1046.

Whitelaw, V.A. & Cowan, J.D. (1981), J. Neurosci., **1**, *12*, 1369-1387.

Yoon, M. (1972), Amer. Zool., **12**, 106.; Exp. Neurol., 37, 451-462; (1976) J. Physiol. Lond., **257**, 621-643.
